# Finite State Automata that Recurrent Cascade-Correlation Cannot Represent

**Stefan C. Kremer**
Department of Computing Science
University of Alberta
Edmonton, Alberta, CANADA   T6H 5B5

## Abstract

This paper relates the computational power of Fahlman's Recurrent Cascade Correlation (RCC) architecture to that of finite state automata (FSA).  While some recurrent networks are FSA equivalent, RCC is not. The paper presents a theoretical analysis of the RCC architecture in the form of a proof describing a large class of FSA which cannot be realized by RCC.

## 1 INTRODUCTION

Recurrent networks can be considered to be defined by two components:  a network architecture, and a learning rule.  The former describes how a network with a given set of weights and topology computes its output values, while the latter describes how the weights (and possibly topology) of the network are updated to fit a specific problem.  It is possible to evaluate the computational power of a network architecture by analyzing the types of computations a network could perform assuming appropriate connection weights (and topology).  This type of analysis provides an upper bound on what a network can be expected to learn, since no system can learn what it cannot represent.

Many recurrent network architectures have been proven to be finite state automaton or even Turing machine equivalent (see for example [Alon, 1991], [Goudreau, 1994], [Kremer, 1995], and [Siegelmann, 1992]).  The existence of such equivalence proofs naturally gives confidence in the use of the given architectures.

This paper relates the computational power of Fahlman's Recurrent Cascade Correlation architecture [Fahlman, 1991] to that of finite state automata.  It is organized as follows: Section 2 reviews the RCC architecture as proposed by Fahlman.  Section 3 describes finite state automata in general and presents some specific automata which will play an important role in the discussions which follow.  Section 4 describes previous work by other

authors evaluating RCC's computational power. Section 5 expands upon the previous work, and presents a new class of automata which cannot be represented by RCC. Section 6 further expands the result of the previous section to identify an infinite number of other unrealizable classes of automata. Section 7 contains some concluding remarks.

## 2  THE RCC ARCHITECTURE

The RCC architecture consists of three types of units: input units, hidden units and output units. After training, a RCC network performs the following computation: First, the activation values of the hidden units are initialized to zero. Second, the input unit activation values are initialized based upon the input signal to the network. Third, each hidden unit computes its new activation value. Fourth, the output units compute their new activations. Then, steps two through four are repeated for each new input signal.

The third step of the computation, computing the activation value of a hidden unit, is accomplished according to the formula:

$$a_j(t+1) = \sigma\left( \sum_{i=1}^{i-1} W_{ij}a_i(t+1) + W_{jj}a_j(t) \right).$$

Here, $a_i(t)$ represents the activation value of unit $i$ at time $t$, $\sigma(\bullet)$ represents a sigmoid squashing function with finite range (usually from 0 to 1), and $W_{ij}$ represents the weight of the connection from unit $i$ to unit $j$. That is, each unit computes its activation value by multiplying the new activations of all lowered numbered units and its own previous activation by a set of weights, summing these products, and passing the sum through a logistic activation function. The recurrent weight $W_{jj}$ from a unit to itself functions as a sort of memory by transmitting a modulated version of the unit's old activation value.

The output units of the RCC architecture can be viewed as special cases of hidden units which have weights of value zero for all connections originating from other output units. This interpretation implies that any restrictions on the computational powers of general hidden units will also apply to the output units. For this reason, we shall concern ourselves exclusively with hidden units in the discussions which follow.

Finally, it should be noted that since this paper is about the representational power of the RCC architecture, its associated learning rule will not be discussed here. The reader wishing to know more about the learning rule, or requiring a more detailed description of the operation of the RCC architecture, is referred to [Fahlman, 1991].

## 3  FINITE STATE AUTOMATA

A Finite State Automaton (FSA) [Hopcroft, 1979] is a formal computing machine defined by a 5-tuple $M=(Q,\Sigma,\delta,q_0,F)$, where $Q$ represents a finite set of states, $\Sigma$ a finite input alphabet, $\delta$ a state transition function mapping $Q \times \Sigma$ to $Q$, $q_0 \in Q$ the initial state, and $F \subset Q$ a set of final or accepting states. FSA accept or reject strings of input symbols according to the following computation: First, the FSA's current state is initialized to $q_0$. Second, the next inut symbol of the str ing, selected from $\Sigma$, is presented to the automaton by the outside world. Third, the transition function, $\delta$, is used to compute the FSA's new state based upon the input symbol, and the FSA's previous state. Fourth, the acceptability of the string is computed by comparing the current FSA state to the set of valid final states, $F$. If the current state is a member of $F$ then the automaton is said to accept the string of input symbols presented so far. Steps two through four are repeated for each input symbol presented by the outside world. Note that the steps of this computation mirror the steps of an RCC network's computation as described above.

It is often useful to describe specific automata by means of a *transition diagram* [Hopcroft, 1979]. Figure 1 depicts the transition diagrams of five FSA. In each case, the states, $Q$,

are depicted by circles, while the transitions defined by δ are represented as arrows from the old state to the new state labelled with the appropriate input symbol. The arrow labelled "Start" indicates the initial state, $q_0$; and final accepting states are indicated by double circles.

We now define some terms describing particular FSA which we will require for the following proof. The first concerns input signals which oscillate. Intuitively, the input signal to a FSA oscillates if every $p^{th}$ symbol is repeated for $p > 1$. More formally, a sequence of input symbols, $s(t)$, $s(t+1)$, $s(t+2)$, ..., oscillates with a period of $p$ if and only if $p$ is the minimum value such that: $\forall t \; s(t) = s(t+p)$.

Our second definition concerns oscillations of a FSA's internal state, when the machine is presented a certain sequence of input signals. Intuitively, a FSA's internal state can oscillate in response to a given input sequence if there is some starting state for which every subsequent $\omega^{th}$ state is repeated. Formally, a FSA's state can oscillate with a period of $\omega$ in response to a sequence of input symbols, $s(t)$, $s(t+1)$, $s(t+2)$, ..., if and only if $\omega$ is the minimum value for which:

$$\exists q_0 \text{ s.t. } \forall t \; \delta(q_0, s(t)) = \delta( \; ... \; \delta( \; \delta( \; \delta(q_0, s(t)), s(t+1)), s(t+2)), \; ... \; , s(t+\omega))$$

The recursive nature of this formulation is based on the fact a FSA's state depends on its previous state, which in turn depends on the state before, etc..

We can now apply these two definitions to the FSA displayed in Figure 1. The automaton labelled "a)" has a state which oscillates with a period of $\omega = 2$ in response to any sequence consisting of 0s and 1s (e.g. "00000...", "11111....", "010101...", etc.). Thus, we can say that it has a state cycle of period $\omega = 2$ (i.e. $q_0 q_1 q_0 q_1 ...$), when its input cycles with a period of $p = 1$ (i.e. "0000..."). Similarly, when automaton b)'s input cycles with period $p = 1$ (i.e. "000000..."), its state will cycle with period $\omega = 3$ (i.e. $q_0 q_1 q_2 q_0 q_1 q_2 ...$).

For automaton c), things are somewhat more complicated. When the input is the sequence "0000...", the state sequence will either be $q_0 q_0 q_0 q_0 ...$ or $q_1 q_1 q_1 q_1 ...$ depending on the initial state. On the other hand, when the input is the sequence "1111...", the state sequence will alternate between $q_0$ and $q_1$. Thus, we say that automaton c) has a state cycle of $\omega = 2$ when its input cycles with period $p = 1$. But, this automaton can also have larger state cycles. For example, when the input oscillates with a period $p = 2$ (i.e. "01010101..."), then the state of the automaton will oscillate with a period $\omega = 4$ (i.e. $q_0 q_0 q_1 q_1 q_0 q_0 q_1 q_1 ...$). Thus, we can also say that automaton c) has a state cycle of $\omega = 4$ when its input cycles with period $p = 2$.

The remaining automata also have state cycles for various input cycles, but will not be discussed in detail. The importance of the relationship between input period ($p$) and the state period ($\omega$) will become clear shortly.

## 4 PREVIOUS RESULTS CONCERNING THE COMPUTATIONAL POWER OF RCC

The first investigation into the computational powers of RCC was performed by Giles et. al. [Giles, 1995]. These authors proved that the RCC architecture, regardless of connection weights and number of hidden units, is incapable of representing any FSA which "for the same input has an output period greater than 2" (p. 7). Using our oscillation definitions above, we can re-express this result as: if a FSA's input oscillates with a period of $p = 1$ (i.e. input is constant), then its state can oscillate with a period of at most $\omega = 2$. As already noted, Figure 1b) represents a FSA whose state oscillates with a period of $\omega = 3$ in response to an input which oscillates with a period of $p = 1$. Thus, Giles et. al.'s theorem proves that the automaton in Figure 1b) cannot be implemented (and hence learned) by a RCC network.

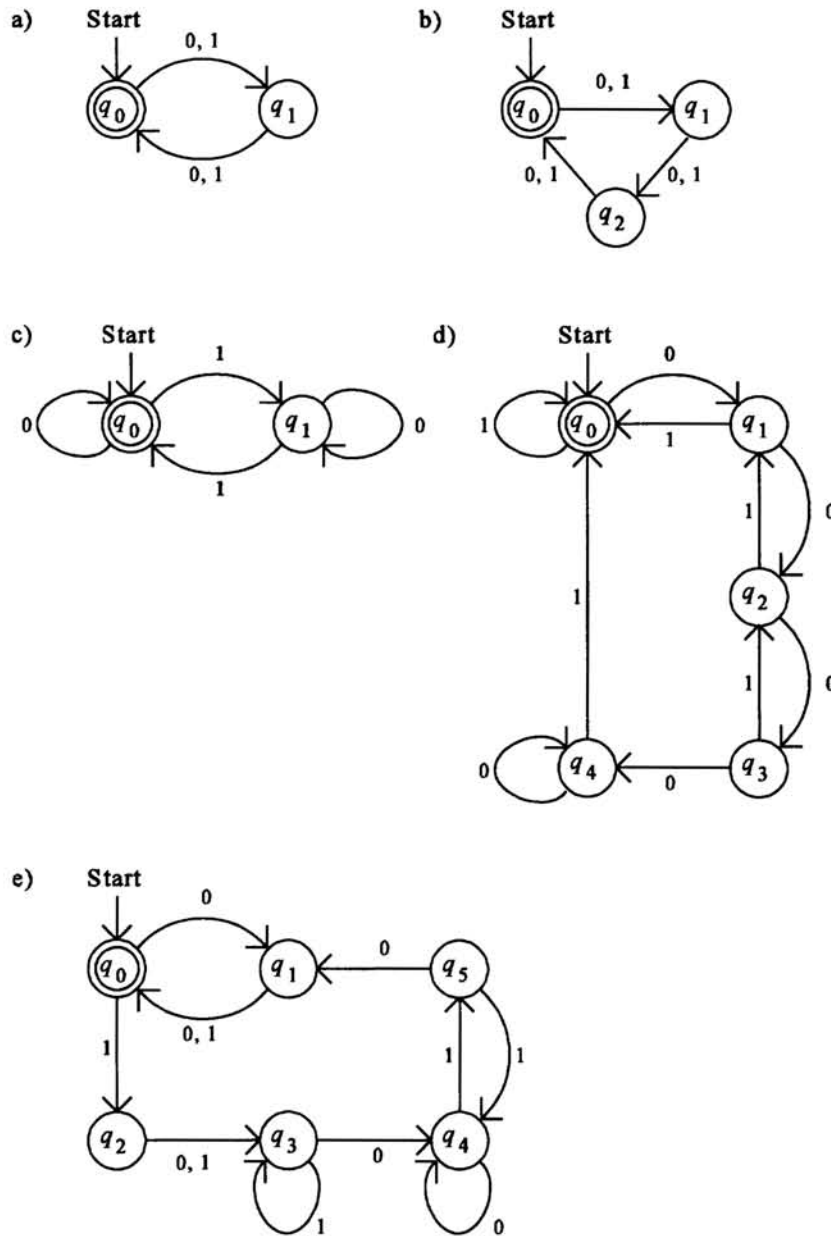

Figure 1: Finite State Automata.

Giles et. al. also examined the automata depicted in Figures 1a) and 1c). However, unlike the formal result concerning FSA b), the authors' conclusions about these two automata were of an empirical nature. In particular, the authors noted that while automata which oscillated with a period of 2 under constant input (i.e. Figure 1a)) were realizable, the automaton of 1c) appeared not be be realizable by RCC. Giles et. al. could not account for this last observation by a formal proof.

## 5 AUTOMATA WITH CYCLES UNDER ALTERNATING INPUT

We now turn our attention to the question: why is a RCC network unable to learn the automaton of 1c)? We answer this question by considering what would happen if 1c) were realizable. In particular, suppose that the input units of a RCC network which implements automaton 1c) are replaced by the hidden units of a RCC network implementing 1a). In this situation, the hidden units of 1a) will oscillate with a period of 2 under constant input. But if the inputs to 1c) oscillate with a period of 2, then the state of 1c) will oscillate with a period of 4. Thus, the combined network's state would oscillate with a period of four under constant input. Furthermore, the cascaded connectivity scheme of the RCC architecture implies that a network constructed by treating one network's hidden units as the input units of another, would not violate any of the connectivity constraints of RCC. In other words, if RCC could implement the automaton of 1c), then it would also be able to implement a network which oscillates with a period of 4 under constant input. Since Giles et. al. proved that the latter cannot be the case, it must also be the case that RCC cannot implement the automaton of 1c).

The line of reasoning used here to prove that the FSA of Figure 1c) is unrealizable can also be applied to many other automata. In fact, any automaton whose state oscillates with a period of more than 2 under input which oscillates with a period 2, could be used to construct one of the automata proven to be illegal by Giles. This implies that RCC cannot implement any automaton whose state oscillates with a period of greater than $\omega = 2$ when its input oscillates with a period of $p = 2$.

## 6 AUTOMATA WITH CYCLES UNDER OSCILLATING INPUT

Giles et. al.'s theorem can be viewed as defining a class of automata which cannot be implemented by the RCC architecture. The proof in Section 5 adds another class of automata which also cannot be realized. More precisely, the two proofs concern inputs which oscillate with periods of one and two respectively. It is natural to ask whether further proofs for state cycles can be developed when the input oscillates with a period of greater than two. We now present the central theorem of this paper, a unified definition of unrealizable automata:

***Theorem***: If the input signal to a RCC network oscillates with a period, $p$, then the network can represent only those FSA whose outputs form cycles of length $\omega$, where $p \bmod \omega = 0$ if $p$ is even and $2p \bmod \omega = 0$ if $p$ is odd.

To prove this theorem we will first need to prove a simpler one relating the rate of oscillation of the input signal to one node in an RCC network to the rate of oscillation of that node's output signal. By "the input signal to one node" we mean the weighted sum of all activations of all connected nodes (i.e. all input nodes, and all lower numbered hidden nodes), but not the recurrent signal. I.e.:

$$\lambda(t+1) = \sum_{i=1}^{j-1} W_{ij} a_i(t+1) .$$

Using this definition, it is possible to rewrite the equation to compute the activation of node $j$ (given in Section 2) as:

$$a_j(t+1) = \sigma(\lambda(t+1) + W_{jj} a_j(t)) .$$

But if we assume that the input signal oscillates with a period of $p$, then every value of $\lambda(t+1)$ can be replaced by one of a finite number of input signals ($\lambda_0, \lambda_1, \lambda_2, \ldots \lambda_{p-1}$). In other words, $\lambda(t+1) = \lambda_{t \bmod p}$. Using this substitution, it is possible to repeatedly expand the addend of the previous equation to derive the formula:

$$a_j(t+1) = \sigma(\lambda_{t \bmod p} + W_{jj} \cdot$$
$$\sigma(\lambda_{(t-1) \bmod p} + W_{jj} \cdot$$
$$\sigma(\lambda_{(t-2) \bmod p} + W_{jj} \cdot \ldots \sigma(\lambda_{(t-p+1) \bmod p} + W_{jj} \cdot a_j(t-p+1)) \ldots )))$$

The unravelling of the recursive equation now allows us to examine the relationship between $a_j(t+1)$ and $a_j(t-p+1)$. Specifically, we note that if $W_{jj} > 0$ or if $p$ is even then $a_j(t+1) = f(a_j(t-p+1))$ implies that $f$ is a monotonically increasing function. Furthermore, since $\sigma$ is a function with finite range, $f$ must also have finite range.

It is well known that for any monotonically increasing function with finite range, $f$, the sequence, $f(x)$, $f(f(x))$, $f(f(f(x)))$, ..., is guaranteed to monotonically approach a fixed point (where $f(x) = x$). This implies that the sequence, $a_j(t+1)$, $a_j(t+p+1)$, $a_j(t+2p+1)$, ..., must also monotonically approach a fixed point (where $a_j(t+1) = a_j(t-p+1)$). In other words, the sequence does **not** oscillate. Since every $p^{\text{th}}$ value of $a_j(t)$ approaches a fixed point, the sequence $a_j(t)$, $a_j(t+1)$, $a_j(t+2)$, ... can have a period of at most $p$, and must have a period which divides $p$ evenly. We state this as our first lemma:

***Lemma 1:*** If $\lambda(t)$ oscillates with even period, $p$, or if $W_{jj} > 0$, then state unit $j$'s activation value must oscillate with a period $\omega$, where $p \bmod \omega = 0$.

We must now consider the case where $W_{jj} < 0$ and $p$ is odd. In this case, $a_j(t+1) = f(a_j(t-p+1))$ implies that $f$ is a monotonically decreasing function. But, in this situation the function $f^2(x) = f(f(x))$ must be monotonically increasing with finite range. This implies that the sequence: $a_j(t+1)$, $a_j(t+2p+1)$, $a_j(t+4p+1)$, ..., must monotonically approach a fixed point (where $a_j(t+1) = a_j(t-2p+1)$). This in turn implies that the sequence $a_j(t)$, $a_j(t+1)$, $a_j(t+2)$, ..., can have a period of at most $2p$, and must have a period which divides $2p$ evenly. Once again, we state this result in a lemma:

***Lemma 2:*** If $\lambda(t)$ oscillates with odd period $p$, and if $W_{jj} < 0$, then state unit $j$ must oscillate with a period $\omega$, where $2p \bmod \omega = 0$.

Lemmas 1 and 2 relate the rate of oscillation of the weighted sum of input signals and lower numbered unit activations, $\lambda(t)$ to that of unit $j$. However, the theorem which we wish to prove relates the rate of oscillation of **only** the RCC network's input signal to the **entire** hidden unit activations. To prove the theorem, we use a proof by induction on the unit number, $i$:

***Basis:*** Node $i=1$ is connected only to the network inputs. Therefore, if the input signal oscillates with period $p$, then node $i$ can only oscillate with period $\omega$, where $p \bmod \omega = 0$ if $p$ is even and $2p \bmod \omega = 0$ if $p$ is odd. (This follows from Lemmas 1 and 2).

***Assumption:*** If the input signal to the network oscillates with period $p$, then node $i$ can only oscillate with period $\omega$, where $p \bmod \omega = 0$ if $p$ is even and $2p \bmod \omega = 0$ if $p$ is odd.

***Proof:*** If the Assumption holds for all nodes $i$, then Lemmas 1 and 2 imply that it must also hold for node $i+1$. $\square$

This proves the theorem:

***Theorem:*** If the input signal to a RCC network oscillates with a period, $p$, then the network can represent only those FSA whose outputs form cycles of length $\omega$, where $p \bmod \omega = 0$ if $p$ is even and $2p \bmod \omega = 0$ if $p$ is odd.

## 7  CONCLUSIONS

It is interesting to note that both Giles et. al.'s original proof and the constructive proof by contradiction described in Section 5 are special cases of the theorem. Specifically, Giles et. al.'s original proof concerns input cycles of length $p=1$. Applying the theorem of Section 6 proves that an RCC network can only represent those FSA whose state transitions form cycles of length $\omega$, where $2(1) \bmod \omega = 0$, implying that state cannot oscillate with a period of greater than 2. This is exactly what Giles et. al concluded, and proves that (among others) the automaton of Figure 1b) cannot be implemented by RCC.

Similarly, the proof of Section 5 concerns input cycles of length $p=2$. Applying our theorem proves that an RCC network can only represent those machines whose state transitions form cycles of length $\omega$, where $(2) \bmod \omega = 0$. This again implies that state cannot oscillate with a period greater than 2, which is exactly what was proven in Section 5. This proves that the automaton of Figure 1c) (among others) cannot be implemented by RCC.

In addition to unifying both the results of Giles et. al. and Section 5, the theorem of Section 6 also accounts for many other FSA which are not representable by RCC. In fact, the theorem identifies an infinite number of other classes of non-representable FSA (for $p=3, p=4, p=5, ...$). Each class itself of course contains an infinite number of machines. Careful examination of the automaton illustrated in Figure 1d) reveals that it contains a state cycle of length 9 ($q_0 q_1 q_2 q_1 q_2 q_3 q_2 q_3 q_4 q_0 q_1 q_2 q_1 q_2 q_3 q_2 q_3 q_4 ...$) in response to an input cycle of length 3 ("001001..."). Since this is not one of the allowable input/state cycle relationships defined by the theorem, it can be concluded that the automaton of Figure 1d) (among others) cannot be represented by RCC.

Finally, it should be noted that it remains unknown if the classes identified by this paper's theorem represent the complete extent of RCC's computational limitations. Consider for example the automaton of Figure 1e). This device has no input/state cycles which violate the theorem, thus we cannot conclude that it is unrepresentable by RCC. Of course, the issue of whether or not this particular automaton is representable is of little interest. However, the class of automata to which the theorem does not apply, which includes automaton 1e), requires further investigation. Perhaps all automata in this class are representable; perhaps there are other subclasses (not identified by the theorem) which RCC cannot represent. This issue will be addressed in future work.

## References

N. Alon, A. Dewdney, and T. Ott, Efficient simulation of finite automata by neural nets, *Journal of the Association for Computing Machinery*, 38 (2) (1991) 495-514.

S. Fahlman, The recurrent cascade-correlation architecture, in: R. Lippmann, J. Moody and D. Touretzky, Eds., *Advances in Neural Information Processing Systems 3* (Morgan Kaufmann, San Mateo, CA, 1991) 190-196.

C.L. Giles, D. Chen, G.Z. Sun, H.H. Chen, Y.C. Lee, and M.W. Goudreau, Constructive Learning of Recurrent Neural Networks: Limitations of Recurrent Cascade Correlation and a Simple Solution, *IEEE Transactions on Neural Networks*, 6 (4) (1995) 829-836.

M. Goudreau, C. Giles, S. Chakradhar, and D. Chen, First-order v.s. second-order single layer recurrent neural networks, *IEEE Transactions on Neural Networks*, 5 (3) (1994) 511-513.

J.E. Hopcroft and J.D. Ullman, *Introduction to Automata Theory, Languages and Computation* (Addison-Wesley, Reading, MA, 1979).

S.C. Kremer, On the Computational Power of Elman-style Recurrent Networks, *IEEE Transactions on Neural Networks*, 6 (4) (1995) 1000-1004.

H.T. Siegelmann and E.D. Sontag, On the Computational Power of Neural Nets, in: *Proceedings of the Fifth ACM Workshop on Computational Learning Theory*, (ACM, New York, NY, 1992) 440-449.